# Synaptic Weight Noise During MLP Learning Enhances Fault-Tolerance, Generalisation and Learning Trajectory

**Alan F. Murray**
Dept. of Electrical Engineering
Edinburgh University
Scotland

**Peter J. Edwards**
Dept. of Electrical Engineering
Edinburgh University
Scotland

## Abstract

We analyse the effects of analog noise on the synaptic arithmetic during MultiLayer Perceptron training, by expanding the cost function to include noise-mediated penalty terms. Predictions are made in the light of these calculations which suggest that fault tolerance, generalisation ability and learning trajectory should be improved by such noise-injection. Extensive simulation experiments on two distinct classification problems substantiate the claims. The results appear to be perfectly general for all training schemes where weights are adjusted incrementally, and have wide-ranging implications for all applications, particularly those involving "inaccurate" analog neural VLSI.

## 1 Introduction

This paper demonstrates both by consideration of the cost function and the learning equations, and by simulation experiments, that injection of random noise on to MLP weights during learning enhances fault-tolerance without additional supervision. We also show that the nature of the hidden node states and the learning trajectory is altered fundamentally, in a manner that improves training times and learning quality. The enhancement uses the mediating influence of noise to distribute information optimally across the existing weights.

Taylor[Taylor, 72] has studied noisy synapses, largely in a biological context, and infers that the noise might assist learning. We have already demonstrated that noise injection both reduces the learning time and improves the network's generalisation ability[Murray, 91],[Murray, 92].  It is established[Matsuoka, 92],[Bishop, 90] that adding noise to the *training data* in neural (MLP) learning improves the "quality" of learning, as measured by the trained network's ability to generalise.  Here we infer (synaptic) noise-mediated terms that sculpt the error function to favour faster learning, and that generate more robust internal representations, giving rise to better generalisation and immunity to small variations in the characteristics of the test data.  Much closer to the spirit of this paper is the work of Hanson[Hanson, 90]. His stochastic version of the delta rule effectively adapts weight means and standard deviations.  Also Sequin and Clay[Sequin, 91] use stuck-at faults during training which imbues the trained network with an ability to withstand such faults.  They also note, but do not pursue, an increased generalisation ability.

This paper presents an outline of the mathematical predictions and verification simulations. A full description of the work is given in [Murray, 93].

## 2    Mathematics

Let us analyse an MLP with $I$ input, $J$ hidden and $K$ output nodes, with a set of $P$ training input vectors $\underline{o}_p = \{o_{ip}\}$, looking at the effect of noise injection into the error function itself. We are thus able to infer, from the additional terms introduced by noise, the characteristics of solutions that tend to **reduce** the error, and those which tend to **increase** it. The former will clearly be favoured, or at least stabilised, by the additional terms, while the latter will be de-stabilised.

Let each weight $T_{ab}$ be augmented by a random noise source, such that $T_{ab} \rightarrow T_{ab} + \Delta_{ab}T_{ab}$, for all weights $\{T_{ab}\}$. Neuron thresholds are treated in precisely the same way. Note in passing, but importantly, that this **synaptic** noise is **not** the same as noise on the input data. Input noise is correlated across the synapses leaving an input node, while the synaptic noise that forms the basis of this study is not. The effect is thus quite distinct.

Considering, therefore, an error function of the form :-

$$\epsilon_{tot,p} = \frac{1}{2}\sum_{k=0}^{K-1}\epsilon_{kp}{}^2 = \frac{1}{2}\sum_{k=0}^{K-1}(o_{kp}(\{T_{ab}\}) - \tilde{o}_{kp})^2 \tag{1}$$

Where $\tilde{o}_{kp}$ is the target output. We can now perform a Taylor expansion of the output $o_{kp}$ to second order, around the noise-free weight set, $\{T_N\}$, and thus augment the error function :-

$$o_{kp} \rightarrow o_{kp} + \sum_{ab}T_{ab}\Delta_{ab}\left(\frac{\partial o_{kp}}{\partial T_{ab}}\right) + \frac{1}{2}\sum_{ab,cd}T_{ab}\Delta_{ab}T_{cd}\Delta_{cd}\left(\frac{\partial^2 o_{kp}}{\partial T_{ab}\partial T_{cd}}\right) + O(>3) \tag{2}$$

If we ignore terms of order $\Delta^3$ and above, and taking the time average over the learning phase, we can infer that two terms are added to the error function :-

$$<\epsilon_{tot}>=<\epsilon_{tot}(\{T_N\})> +\frac{1}{2P}\sum_{p=1}^{P}\sum_{k=0}^{K-1}\Delta^2\sum_{ab}T_{ab}{}^2\left[\left(\frac{\partial o_{kp}}{\partial T_{ab}}\right)^2 + \epsilon_{kp}\left(\frac{\partial^2 o_{kp}}{\partial T_{ab}{}^2}\right)\right] \tag{3}$$

Consider also the perceptron rule update on the hidden-output layer along with the expanded error function :-

$$< \delta T_{kj} >= -\tau \sum_p < \epsilon_{kp} o_{jp} o'_{kp} > -\tau \frac{\Delta^2}{2} \sum_P < o_{jp} o'_{kp} > \times \sum_{ab} T_{ab}^2 \frac{\partial^2 o_{kp}}{\partial T_{ab}^2} \quad (4)$$

averaged over several training epochs (which is acceptable for small values of $\tau$ the *adaption rate* parameter).

## 3   Simulations

The simulations detailed below are based on the *virtual targets* algorithm [Murray, 92], a variant on backpropagation, with broadly similar performance. The "targets" algorithm was chosen for its faster convergence properties. Two contrasting classification tasks were selected to verify the predictions made in the following section by simulation. The first, a feature location task, uses *real world* normalised greyscale image data. The task was to locate eyes in facial images - to classify sections of these as either "eye" or "not-eye". The network was trained on $16 \times 16$ preclassified sections of the images, classified as eyes and not-eyes. The not-eyes were random sections of facial images, avoiding the eyes (see Fig. 1). The second,

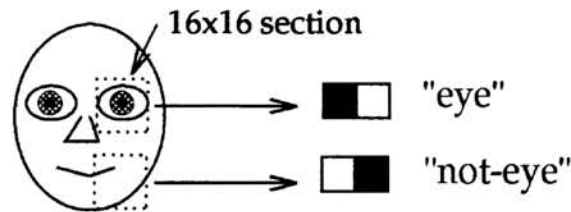

Figure 1: The eye/not-eye classifier.

a more artificial task, was the ubiquitous character encoder (Fig. 2) where a 25-

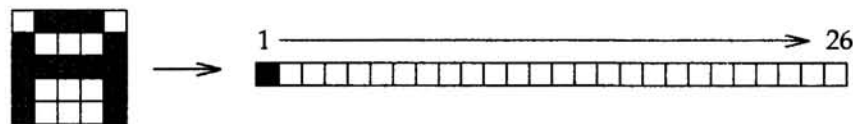

Figure 2: The character encoder task.

dimensional binary input vector describing the 26 alphabetic characters (each $5 \times 5$ pixels) was used to train the network with a one-out-of-26 output code.

During the simulations noise was added to the weights at a level proportional to the weight size and at a probability distribution of uniform density (i.e. $-\Delta_{max} < \Delta < \Delta_{max}$). Levels of up to 40% were probed in detail - although it is clear that the expansion above is not quantitatively valid at this level. Above these percentages further improvements were seen in the network performance, although the dynamics of the training algorithm became chaotic. The injected noise level was reduced

smoothly to a minimum value of 1% as the network approached convergence (as evidenced by the highest output bit error). As in all neural network simulations, the results depended upon the training parameters, network sizes and the random start position of the network. To overcome these factors and to achieve a meaningful result 35 weight sets were produced for each noise level. All other characteristics of the training process were held constant. The results are therefore not simply pathological freaks.

## 4    Prediction/Verification

### 4.1    Fault Tolerance

Consider the first derivative penalty term in the expanded cost function (3), averaged over all patterns, output nodes and weights :-

$$K \times \Delta^2 \left[ T_{ab}{}^2 \left( \frac{\partial o_{kp}}{\partial T_{ab}} \right)^2 \right] \tag{5}$$

The implications of this term are straightforward. For large values of the (weighted) average magnitude of the derivative, the overall error is increased. This term therefore causes solutions to be favoured where the dependence of outputs on individual weights is evenly distributed across the entire weight set. Furthermore, weight saliency should not only have a lower average value, but a smaller scatter across the weight set as the training process attempts to reconcile the competing pressures to reduce both (1) and (5). This more distributed representation should be manifest in an improved tolerance to faulty weights.

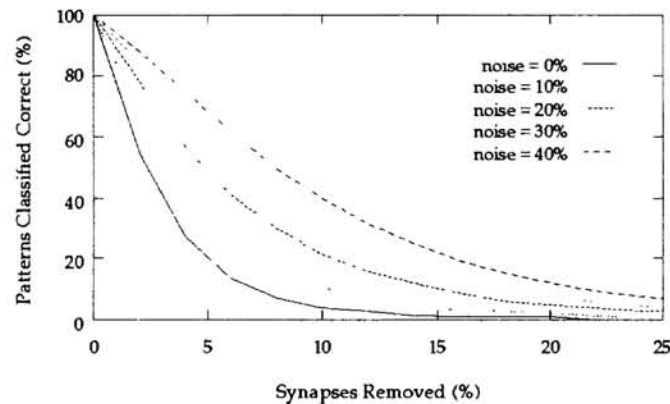

Figure 3: Fault tolerance in the character encoder problem.

Simulations were carried out on 35 weight sets produced for each of the two problems at each of 5 levels of noise injected during training. Weights were then randomly removed and the networks tested on the training data. The resulting graphs (Fig. 3, 4) show graceful degradation with an increased tolerance to faults with injected noise during training. The networks were highly constrained for these simulations to remove some of the natural redundancy of the MLP structure. Although the eye/not-eye problem contains a high proportion of redundant information, the

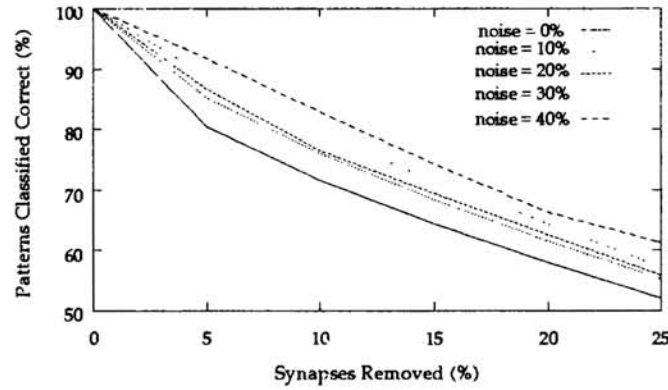

Figure 4: Fault tolerance enhancement in the eye/not-eye classifier.

improvement in the networks ability to withstand damage, with injected noise, is clear.

## 4.2   Generalisation Ability

Considering the derivative in equation 5, and looking at the input-hidden weights. The term that is added to the error function, again averaged over all patterns, output nodes and weights is :-

$$K \times \Delta^2 \left[ T_{ji}^{2} \; o_{kp}'^{2} \; T_{kj}^{2} \; o_{jp}'^{2} \; o_{ip}^{2} \right] \tag{6}$$

If an output neuron has a non-zero connection from a particular hidden node ($T_{kj} \neq 0$), and provided the input $o_{ip}$ is non-zero and is connected to the hidden node ($T_{ji} \neq 0$), there is also a term $o_{jp}'$ that will **tend to favour solutions with the hidden nodes also turned firmly ON or OFF** (i.e. $o_{jp} = 0 \; or \; 1$). Remembering, of course, that all these terms are noise-mediated, and that during the early stages of training, the "actual" error $\epsilon_{kp}$, in (1), will dominate, this term will de-stabilise final solutions that balance the hidden nodes on the slope of the sigmoid. Naturally, hidden nodes $o_j$ that are firmly ON or OFF are less likely to change state as a result of small variations in the input data $\{o_i\}$. This should become evident in an increased tolerance to input perturbations and therefore an increased generalisation ability.

Simulations were again carried out on the two problems using 35 weight sets for each level of injected synaptic noise during training. For the character encoder problem generalisation is not really an issue, but it is possible to verify the above prediction by introducing random gaussian noise into the input data and noting the degradation in performance. The results of these simulations are shown in Fig. 5, and clearly show an increased ability to withstand input perturbation, with injected noise into the synapses during training.

Generalisation ability for the eye/not-eye problem is a real issue. This problem therefore gives a valid test of whether the synaptic noise technique actually improves generalisation performance. The networks were therefore tested on previously unseen facial images and the results are shown in Table 1. These results show

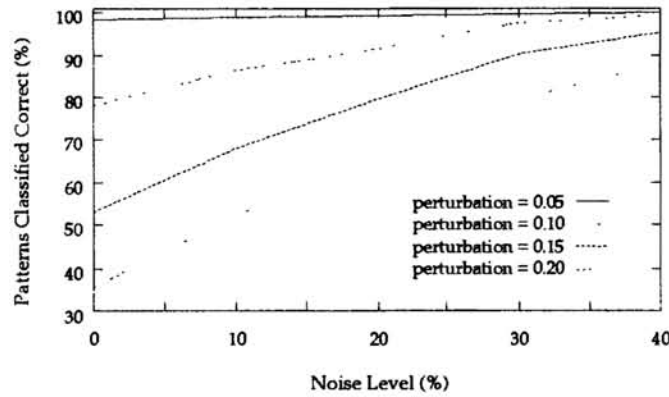

Figure 5: Generalisation enhancement shown through increased tolerance to input perturbation, in the character encoder problem.

|  | Correctly Classified (%) | | | | |
|---|---|---|---|---|---|
| Noise Levels | 0% | 10% | 20% | 30% | 40% |
| Test Patterns | 67.875 | 70.406 | 70.416 | 72.454 | 75.446 |

Table 1: Generalisation enhancement shown through increased ability to classifier previously unseen data, in the eye/not-eye task.

dramatically improved generalisation ability with increased levels of injected synaptic noise during training. An improvement of approximately 8% is seen - consistent with earlier results on a different "real" problem [Murray, 91].

## 4.3   Learning Trajectory

Considering now the second derivative penalty term in the expanded cost function (2). This term is complex as it involves second order derivatives, and also depends upon the sign and magnitude of the errors themselves $\{\epsilon_{kp}\}$. The simplest way of looking at its effect is to look at a single exemplar term :-

$$K\Delta^2 \, \epsilon_{kp}{T_{ab}}^2 \left(\frac{\partial^2 o_{kp}}{{\partial T_{ab}}^2}\right) \tag{7}$$

This term implies that when the combination of $\epsilon_{kp}\frac{\partial^2 o_{kp}}{\partial T_{ab}^2}$ is negative then the overall cost function error is reduced and *vice versa*. The term (7) is therefore constructive as it can actually lower the error locally via noise injection, whereas (6) always increases it. (7) can therefore be viewed as a sculpting of the error surface during the early phases of training (i.e. when $\epsilon_{kp}$ is substantial). In particular, a weight set with a higher "raw" error value, calculated from (1), may be favoured over one with a lower value if noise-injected terms indicate that the "poorer" solution is located in a promising area of weight space. This "look-ahead" property should lead to an enhanced learning trajectory, perhaps finding a solution more rapidly.

In the augmented weight update equation (4), the noise is acting as a medium projecting statistical information about the character of the entire weight set on to

the update equation for each particular weight. So, the effect of the noise term is to account not only for the weight currently being updated, but to add in a term that estimates what the other weight changes are likely to do to the output, and adjust the size of the weight increment/decrement as appropriate.

To verify this by simulation is not as straightforward as the other predictions. It is however possible to show the mean training time for each level of injected noise. For each noise level, 1000 random start points were used to allow the underlying properties of the training process to emerge. The results are shown in Fig. 6 and

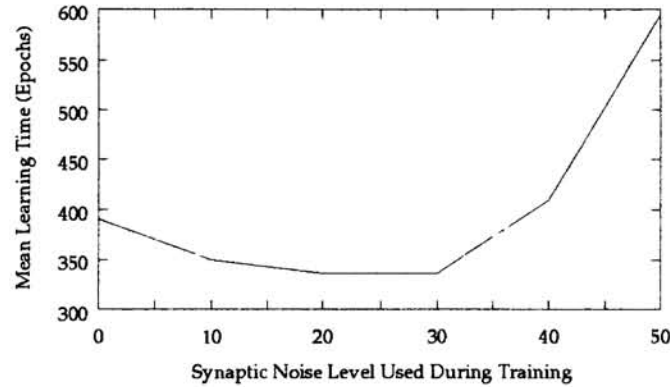

Figure 6: Training time as a function of injected synaptic noise during training.

clearly show that at low noise levels ($\leq 30\%$ for the case of the character encoder) a definite reduction in training times are seen. At higher levels the chaotic nature of the "noisy learning" takes over.

It is also possible to plot the combination of $\epsilon_{kp} \frac{\partial^2 o_{kp}}{\partial T_{ab}^2}$. This is shown in Fig. 7, again for the character encoder problem. The term (7) is reduced more quickly

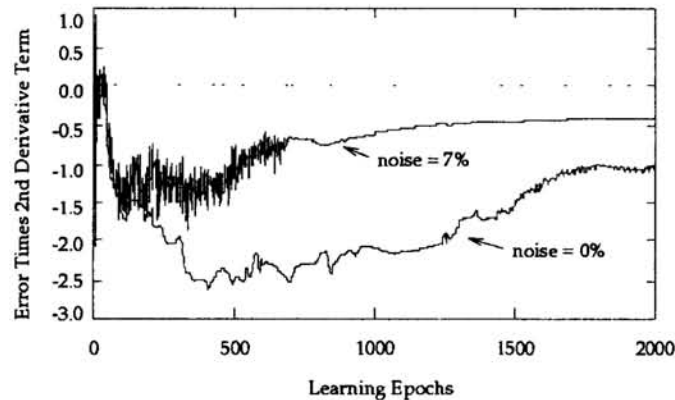

Figure 7: The second derivative × error term trajectory for injected synaptic noise levels 0% and 7%.

with injected noise, thus effecting better weight changes via (4). At levels of noise > 7% the effect is exaggerated, and the noise mediated improvements take place

during the first 100-200 epochs of training. The level of 7% is displayed simply because it is visually clear what is happening, and is also typical.

## 5   Conclusion

We have shown both by mathematical expansion and by simulation that injecting random noise on to the synaptic weights of a MultiLayer Perceptron during the training phase enhances fault-tolerance, generalisation ability and learning trajectory. It has long been held that any inaccuracy during training is detrimental to MLP learning. This paper proves that **analog** inaccuracy is not. The mathematical predictions are perfectly general and the simulations relate to a non-trivial classification task and a "real" world problem. The results are therefore important for the designers of analog hardware and also as a non-invasive technique for producing learning enhancements in the software domain.

### Acknowledgements

We are grateful to the Science and Engineering Research Council for financial support, and to Lionel Tarassenko and Chris Bishop for encouragement and advice.

## References

[Taylor, 72]     J. G. Taylor, "Spontaneous Behaviour in Neural Networks", *J. Theor. Biol.*, vol. 36, pp. 513-528, 1972.

[Murray, 91]     A. F. Murray, "Analog Noise-Enhanced Learning in Neural Network Circuits," *Electronics Letters*, vol. 2, no. 17, pp. 1546-1548, 1991.

[Murray, 92]     A. F. Murray, "Multi-Layer Perceptron Learning Optimised for On-Chip Implementation - a Noise Robust System," *Neural Computation*, vol. 4, no. 3, pp. 366-381, 1992.

[Matsuoka, 92]     K. Matsuoka, "Noise Injection into Inputs in Back-Propagation Learning", *IEEE Trans. Systems, Man and Cybernetics*, vol. 22, no. 3, pp. 436-440, 1992.

[Bishop, 90]     C. Bishop, "Curvature-Driven Smoothing in Backpropagation Neural Networks," *IJCNN*, vol. 2, pp. 749-752, 1990.

[Hanson, 90]     S. J. Hanson, "A Stochastic Version of the Delta Rule", *Physica D*, vol. 42, pp. 265-272, 1990.

[Sequin, 91]     C. H. Sequin, R. D. Clay, "Fault Tolerance in Feed-Forward Artificial Neural Networks", Neural Networks : Concepts, Applications and Implementations, vol. 4, pp. 111-141, 1991.

[Murray, 93]     A. F. Murray, P. J. Edwards, "Enhanced MLP Performance and Fault Tolerance Resulting from Synaptic Weight Noise During Training", *IEEE Trans. Neural Networks*, 1993, In Press.
